# *CSPG*: Crossing Sparse Proximity Graphs for Approximate Nearest Neighbor Search

**Ming Yang, Yuzheng Cai, Weiguo Zheng**
School of Data Science, Fudan University, China
{ yangm24, yuzhengcai21 } @m.fudan.edu.cn  zhengweiguo@fudan.edu.cn

## Abstract

The state-of-the-art approximate nearest neighbor search (ANNS) algorithm builds a large proximity graph on the dataset and performs a greedy beam search, which may bring many unnecessary explorations. We develop a novel framework, namely *corssing sparse proximity graph (CSPG)*, based on random partitioning of the dataset. It produces a smaller sparse proximity graph for each partition and routing vectors that bind all the partitions. An efficient two-staged approach is designed for exploring *CSPG*, with fast approaching and cross-partition expansion. We theoretically prove that *CSPG* can accelerate the existing graph-based ANNS algorithms by reducing unnecessary explorations. In addition, we conduct extensive experiments on benchmark datasets. The experimental results confirm that the existing graph-based methods can be significantly outperformed by incorporating *CSPG*, achieving 1.5x to 2x speedups of *QPS* in almost all recalls.

## 1   Introduction

*Nearest Neighbor Search* (NNS) aims to find some vectors in a set of high-dimensional vectors with the smallest distance to a query vector. It is becoming increasingly popular in various application domains [1–7], such as information retrieval [8, 9], pattern recognition [10, 11], recommendation systems [12, 13], and retrieval augmented generation (RAG) [14, 15]. However, it is costly to find the exact nearest neighbors in practice, thus recent studies have focused on *Approximate Nearest Neighbor Search* (ANNS), which targets efficiency while mildly relaxing accuracy constraints [16, 17].

Existing ANNS algorithms can be divided into four categories [16], including tree-based approaches [1, 18–22], hashing-based approaches [23–27], quantization-based approaches [28–31], and graph-based approaches [6, 32–35]. Among these approaches, graph-based ANNS algorithms stand out with the high answer quality and low latency [16], by constructing a *Proximity Graph (shorted as PG)* on the given vector dataset. As shown in Figure 1, each vector is represented by a node in the graph, and each node is connected to its nearby neighbors.

For the greedy beam search over the proximity graph, it is observed that the distance computation dominates the overall time cost [36, 37]. Since at each step, all neighbors of the current node are pushed into the candidate set according to the computed distance. The number of distance computations can be calculated as $\sigma$ multiplied by the number of explored nodes [6], where $\sigma$ is the average degree of the graph. Intuitively, searching within a smaller graph requires less exploration and thus reduces the overall cost, which has been proved for a particular type of proximity graph, i.e., Monotonic Search Network (MSNET) [6, 38]. However, for most proximity graphs, it is very likely to degrade the answer quality when the graph is smaller.

In this paper, we present a novel and effective framework, namely *Crossing Sparse Proximity Graph (CSPG)*, enabling efficient search while not sacrificing answer quality. The basic idea is to reduce the number of explored vectors by searching in the much smaller graphs. Specifically, we randomly divide the whole dataset into several partitions, and for each partition, we construct a proximity graph that is smaller than the proximity graph built on the whole dataset. These partitions share a set of *routing*

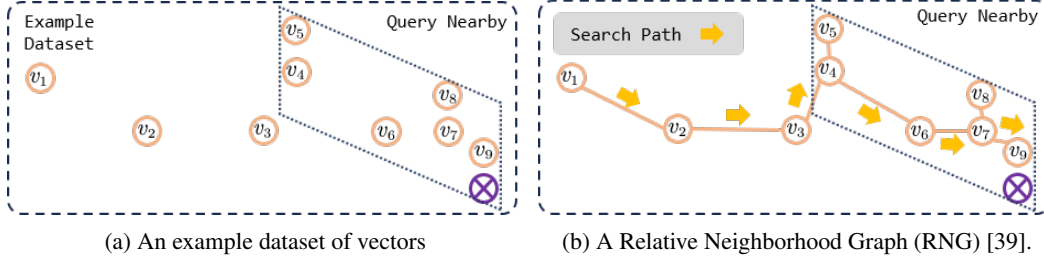

(a) An example dataset of vectors      (b) A Relative Neighborhood Graph (RNG) [39].

Figure 1: An example dataset of vectors and its proximity graph.

*vectors* (Section 3.1) that allow the greedy search to travel across different partitions dynamically. The query process involves two stages, i.e., fast approaching and cross-partition expansion. The first stage conducts the greedy search within one partition, using a small candidate set to quickly approach the nearby regions of the query vector. Then, the second stage continues the greedy search with a larger candidate set, allowing it to travel across different partitions for more precise results.

We theoretically prove that the expected number of explored vectors when searching across these small graphs is the same as searching on the small proximity graph for one of the partitions. Hence, by random partitioning with randomly sampled routing vectors, we can reduce the number of explored vectors compared with the traditional proximity graph built on the whole dataset, thus reducing the number of distance computations. By integrating *CSPG* with various graph-based ANNS algorithms, extensive experiments show that it significantly speeds up the query performance on benchmark datasets, and the detailed empirical results also align with our theoretical analysis.

**Contributions.** In summary, we make the following contributions in this paper.

- To improve the query performance by reducing the number of explored vectors, we propose a general framework, namely *Crossing Sparse Proximity Graph (CSPG)*, through random partitioning and random routing vectors. This framework can integrate with and enhance the existing graph-based ANNS indexes.
- We develop an efficient two-staged search paradigm over the *CSPG*, including fast approaching and cross-partition expansion.
- We theoretically prove that *CSPG* can benefit the existing graph-based ANNS algorithms by introducing *Approximate Monotonic Search Network* (AMSNET) that considers the distance backtracking in the search path.
- Extensive experiments confirm that by integrating the *CSPG*, the existing graph-based algorithms can be speeded up significantly under the same answer quality.

## 2 Background

### 2.1 Problem definition

Let $\mathcal{D} = \{v_1, v_2, ..., v_n\}$ denote the dataset of $n$ vectors, where $v_i$ represents a vector in the $d$-dimensional Euclidean space $\mathbb{R}^d$. The L2 distance between any two vectors $p \in \mathbb{R}^d$ and $q \in \mathbb{R}^d$ is denoted as $\delta(p, q)$. The task of $k$-nearest neighbor search ($k$-NNS) can be defined as follows.

**Definition 1** ($k$-Nearest Neighbor Search, shorted as $k$-NNS)**.** *Given a dataset $\mathcal{D}$ and a query vector $q$, $k$-NNS returns a subset of $k$ vectors, denoted by $T \subseteq \mathcal{D}$, such that for any $t \in T$ and $v \in \mathcal{D} \setminus T$, we have $\delta(v, q) \geq \delta(t, q)$.*

**Definition 2** ($k$-Approximate Nearest Neighbor Search, shorted as $k$-ANNS )**.** *Given a dataset $\mathcal{D}$ and a query vector $q$, $k$-ANNS returns a subset of $k$ vectors, denoted by $S \subseteq \mathcal{D}$, such that $|S \cap T|$ is as large as possible, where $T$ is the answer set to $k$-NNS w.r.t. the query $q$. For simplicity, $k$ is omitted when $k = 1$.*

In other words, the task of $k$-ANNS returns $k$ approximate closest vectors of the query vector, not guaranteeing all the exact top-$k$ nearest vectors, to improve query efficiency.

### 2.2 Graph-based ANNS algorithms

As discussed above, graph-based ANNS algorithms conduct a best-first greedy beam search on the proximity graphs to approach the closest nodes for a query vector. Their built proximity graphs can be classified into four categories [16, 40] as follows. Please refer to Appendix A for more details.

**Delaunay Graph (DG) [41, 42]**. It ensures that for any edge, no other vectors will be situated within the hypersphere defined by an edge connecting two vectors, where the hypersphere is centered at the midpoint of the edge and the length of the edge is the diameter.

**Relative Neighborhood Graph (RNG)** [39]. It guarantees that for any edge between $p$ and $q$, no other vectors will reside within the $lune(p,q) = \{u \in \mathbb{R}^d \,|\, \delta(u,p) \leq \delta(p,q) \wedge \delta(u,q) \leq \delta(p,q)\}$. RNG imposes stricter restrictions on its edges, thus decreasing the average degree [43].

**K-Nearest Neighbor Graph (KNNG)** [44]. In KNNG, neighbors of each vector $v \in \mathcal{D}$ are its top-$k$ nearest neighbors in $\mathcal{D}$. NN-Descent [44] proposes a method for constructing KNNG.

**Minimum Spanning Tree (MST) Graph** [45]. The MST utilizes distances between vectors as edge weights. Then, it performs hierarchical clustering on the dataset multiple times randomly, adding some edges to the edge set. MST can establish global connectivity with a minimal number of edges.

## 2.3 Monotonic Search Network

Monotonic Search Network provides theoretical results to understand the costs of greedy search.

**Definition 3** (Monotonic Path, shorted as MP [38]). *Given a proximity graph built on dataset $\mathcal{D}$, for two nodes $p$ and $u$ in the graph, a path from $p$ to $u$ is denoted as $p \rightsquigarrow u = \{v_1, v_2, ..., v_t\}$, where $p = v_1$ and $u = v_t$. It is a monotonic path iff it satisfies that $\delta(v_1, u) > \delta(v_2, u) > ... > \delta(v_{t-1}, u)$.*

**Definition 4** (Monotonic Search Network, shorted as MSNET [38]). *Given a dataset $\mathcal{D}$ consisting of $n$ vectors in the space $\mathbb{R}^d$, a proximity graph built on $\mathcal{D}$ is called a monotonic search network iff there exists at least one monotonic path from $p$ to $u$ for any two nodes $p$ and $u$ in $\mathcal{D}$.*

When running a greedy beam search in an MSNET, we will continuously approach the query vector since the distance strictly decreases at each step, i.e., distance backtracking can be avoided [38]. Let $C$ denote the smallest convex hull that can cover a set of $n$ $d$-dimensional vectors $\mathcal{D}$, and let $R$ represent the maximum distance between two vectors in $\mathcal{D}$. Denote the volume of $C$ as $V_C$ and let $V_B(\cdot, R)$ represent the volume of a sphere with radius $R$. For an MSNET built on $\mathcal{D}$, when there exists a constant $\kappa$ s.t. $\kappa V_C \geq V_B(\cdot, R)$, which implies that the distribution of vectors should be relatively uniform (never in some extremely special shape), the search length expectation of an MSNET (denoted as $\mathbb{E}^M$) is $\mathcal{O}\left(n^{\frac{1}{d}} \log n^{\frac{1}{d}} / \Delta r\right)$ as proved in [6], where

$$\Delta r = \min_{v_1, v_2, v_3 \in \mathcal{D}} \left\{ \left| \delta\left(v_1, v_2\right) - \delta\left(v_1, v_3\right) \right|, \left| \delta\left(v_1, v_2\right) - \delta\left(v_2, v_3\right) \right|, \left| \delta\left(v_1, v_3\right) - \delta\left(v_2, v_3\right) \right| \right\}.$$

In other words, $\Delta r$ is the minimum distance difference for any non-isosceles triangle on $\mathcal{D}$. As $n$ increases, $\Delta r$ decreases and approaches a constant value when $n$ is large [6].

# 3 Crossing Sparse Proximity Graphs

## 3.1 *CSPG*: Crossing Sparse Proximity Graphs

An effective approach to the $k$-ANNS problem is expected to identify more vectors that are closest to the query with a minimal cost. A straightforward approach is to relax edge selection by allowing a vector to connect with both nearby and relatively distant neighbors. To guarantee efficiency, the node degree cannot be increased too much, making it challenging to maintain both nearby and distant neighbors. To address the problem, we propose a method to maximize the number of vectors searched near the query without increasing average node degrees. The basic idea is randomly partitioning the dataset $\mathcal{D}$ into multiple groups that share some common vectors, called *routing vectors*. Then a sparse proximity graph (shorted SPG) is built for each group of vectors. Since these SPGs are sparser than that built for the whole dataset $\mathcal{D}$, allowing larger steps to approach the query quickly. Moreover, the *routing vectors* across multiple SPGs enable efficient fine-grained search. For ease of presentation, let *PG($\mathcal{D}$)* denote the proximity graph built on the dataset $\mathcal{D}$.

**Definition 5** (Random Partition). *Given a vector dataset $\mathcal{D}$, the group of subsets $\mathcal{P}_1, \mathcal{P}_2, \cdots, \mathcal{P}_m$ constitute a random partition of $\mathcal{D}$ such that (1) $\mathcal{P}_1 \cup \mathcal{P}_2 \cup \cdots \cup \mathcal{P}_m = \mathcal{D}$, (2) $\mathcal{P}_1 \cap \mathcal{P}_2 \cap \cdots \cap \mathcal{P}_m = \mathcal{C}$, and (3) $(\mathcal{P}_i \backslash \mathcal{C}) \cap (\mathcal{P}_i \backslash \mathcal{C}) = \emptyset$ for $i \neq j$, where $\mathcal{P}_i$ is randomly sampled from $\mathcal{D}$ and $\mathcal{C}$ is the common vectors shared by all the subsets (also called routing vectors).*

**Definition 6** (Crossing Sparse Proximity Graphs, shorted as *CSPG*). *Given a vector dataset $\mathcal{D}$, its CSPG($\mathcal{D}$) consists of multiple proximity graphs $\mathcal{G}_1, \mathcal{G}_2, \cdots, \mathcal{G}_m$ for $\mathcal{D}$'s random partition $\mathcal{P}_1, \mathcal{P}_2, \cdots, \mathcal{P}_m$, respectively, i.e., $\mathcal{G}_i = PG(\mathcal{P}_i)$.*

Note that $\mathcal{P}_i$ is sparser than $\mathcal{D}$ as it is randomly sampled from $\mathcal{D}$. Thus, the average edge length (i.e., the distance between two vectors) of the resultant proximity graph $\mathcal{G}_i$ is larger than that of the

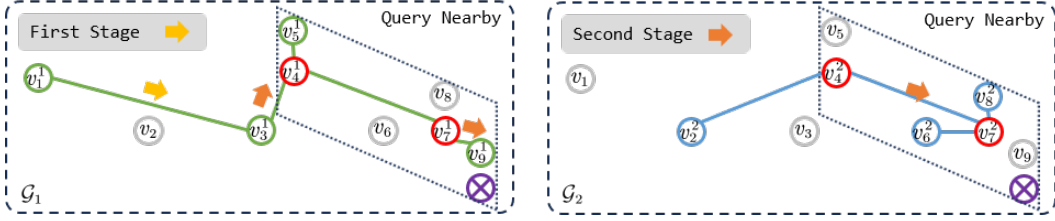

Figure 2: An example of *CSPG* index, where the proximity graphs are built using relative neighborhood graph $\mathcal{G}_1$ and $\mathcal{G}_2$ (with very similar degree to Figure 1b.

proximity graph for $\mathcal{D}$. Generally, any existing graph-based index can be used to build the proximity graphs in *CSPG*. Since the partitions share routing vectors, the corresponding proximity graphs are interrelated through these routing vectors. Hence, the routing vectors serve to navigate the greedy search across different proximity graphs. Let $v_j^i$ denote that vector $v_j$ belongs to graph $\mathcal{G}_i$.

**Example 1.** *For the dataset $\mathcal{D} = \{v_1, v_2, ..., v_9\}$ in Figure 1, we build the CSPG in Figure 2, by randomly sampling 2 routing vectors $\mathcal{C} = \{v_4, v_7\}$. And there are two partitions $\mathcal{P}_1 = \{v_1, v_3, v_4, v_5, v_7, v_9\}$ and $\mathcal{P}_2 = \{v_2, v_4, v_6, v_7, v_8\}$, where routing vectors are highlighted in red, with the green graph representing $\mathcal{G}_1$ and the blue graph representing $\mathcal{G}_2$.*

### 3.2 Novelty of *CSPG*

**Comparison with *Inverted File Index (IVF)*.** Generally, *IVF* uses clustering algorithms (e.g., k-means) to divide the dataset into buckets. During the search, it selects some buckets with the closest centroids w.r.t. the query, after which vectors in such buckets will be scanned for final results. The buckets of *IVF* index disrupt the distribution of vectors in the original dataset. In contrast, *CSPG* preserves the original distribution by random partition, which diffuses all vectors, and the routing vectors are used for connecting all proximity graphs from different partitions.

**Comparison with *PG*($\mathcal{D}$).** The *CSPG* index is built based on random partitions, with the help of routing vectors for connectivity. When the number of partitions $m = 1$, *CSPG* falls into the special case that builds a *PG* index over all vectors, which is consistent with most state-of-the-art graph-based ANNS algorithms. Some existing *PG* index [46, 47] also utilized similar ideas of using data partition and redundancy. Existing studies [46, 47] uses k-means or other methods to divide the dataset, and obtain some redundant vectors. Then, such algorithms also construct proximity graphs in each partition separately, which are eventually *merged into a large PG* as the final proximity graph covering all vectors. Such existing techniques are developed to deal with a huge amount of vectors, making it feasible to handle large datasets. In contrast, the *CSPG* methods aim at building several proximity graphs to speed up query answering.

**Comparison with *HNSW*.** (1) From the perspective of redundancy, the lower level of *HNSW* contains all vectors from the upper level. But in *CSPG*, there is a common overlap between partitions, and the remaining points of different partitions are distinct. (2) From the perspective of structure, *HNSW* is a hierarchical structure. In contrast, *CSPG* serves as a framework rather than a specific structure (hierarchical or flat), allowing to enhance query performance across a broad range of mainstream graph indices. *HNSW* transitions from top level to bottom level unidirectionally, while *CSPG* builds horizontally with smaller, sparser proximity graphs. (3) From the perspective of searching, *HNSW* can only unidirectionally search each level from top to bottom, and the final results are obtained from the bottom-level graph. But *CSPG* can traverse back and forth between different sparse proximity graphs, collecting final results from various partitions. The performance of *HNSW* is closely tied to the quality of the bottom-level graph, while *CSPG* generates more diverse and robust answers by leveraging cross-partition traversal and result collection.

### 3.3 *CSPG* index construction and updates

Algorithm 2 presents the process for *CSPG* index construction. It first samples the routing vectors $RV$ from dataset $\mathcal{D}$, then the other vectors $\mathcal{D}\backslash RV$ are randomly assigned to $m$ partitions. Finally, for each partition $\mathcal{P}_i$, we construct the proximity graph by applying the graph-based ANNS algorithm.

Since *CSPG* is a framework based on mainstream proximity graphs, current updating methods of the underlying graph index are applicable. Moreover, the random partitioning makes it highly flexible for vector insertion and deletion. Details are presented in Algorithms 3 and 4 of Appendix B.

---

**Algorithm 1** Search on *CSPG* index

---

**Require:** *CSPG* index $\mathcal{G} = \{\mathcal{G}_1, \mathcal{G}_2, ..., \mathcal{G}_m\}$, query vector $q$, parameters $ef_1$ and $ef_2$
**Ensure:** $k$ nearest neighbors of $q$
 1: $\mathcal{L} \leftarrow \{$ selected entry vector $p \in \mathcal{P}_1\}$, $visited \leftarrow \{p\}$ &emsp;&emsp;&emsp;&emsp;&emsp;&emsp; ▷ First stage starts
 2: **while** $|\mathcal{L}| \neq 0$ **do**
 3: &emsp; $(r, h) \leftarrow$ the closest vector w.r.t. $q$ in $\mathcal{L}$
 4: &emsp; **for all** unvisited neighbor $u$ of $r$ in $\mathcal{G}_1$ **do**
 5: &emsp;&emsp; $\mathcal{L} \leftarrow \mathcal{L} \cup \{(u, 1)\}$, $visited \leftarrow visited \cup \{u\}$
 6: &emsp; **if** $|\mathcal{L}| > ef_1$ **then** remove the farthest vectors w.r.t. $q$ to keep $|\mathcal{L}| = ef_1$
 7: $p \leftarrow$ the closest vector w.r.t. $q$ in $visited$ &emsp;&emsp;&emsp;&emsp;&emsp;&emsp;&emsp;&emsp; ▷ Second stage starts
 8: $\mathcal{L} \leftarrow \{(p, 1)\}$, $visited \leftarrow \{p\}$
 9: **while** $|\mathcal{L}| \neq 0$ **do**
10: &emsp; $(r, h) \leftarrow$ the closest vector w.r.t. $q$ in $\mathcal{L}$
11: &emsp; **for all** unvisited neighbor $u$ of $r$ in $\mathcal{G}_h$ **do**
12: &emsp;&emsp; $\mathcal{L} \leftarrow \mathcal{L} \cup \{(u, h)\}$, $visited \leftarrow visited \cup \{u\}$
13: &emsp;&emsp; **if** $u$ is a routing vector **then** $\mathcal{L} \leftarrow \mathcal{L} \cup \{(u, i) \mid i \in \{1, 2, ..., m\} \wedge i \neq h\}$
14: &emsp; **if** $|\mathcal{L}| > ef_2$ **then** remove the farthest vectors w.r.t. $q$ to keep $|\mathcal{L}| = ef_2$
15: **return** top-$k$ closest vectors w.r.t. $q$ in $visited$

---

**Time and space complexity.** For dataset $\mathcal{D}$ with $n$ vectors, the time and space cost of building index are $\mathcal{O}(T(n))$ and $\mathcal{O}(S(n))$, respectively. As each partition has $\frac{n(1-\lambda)}{m} + \lambda n$ vectors, *CSPG* consumes $\mathcal{O}\left(m \cdot T\left(\frac{n(1-\lambda)}{m} + \lambda n\right)\right)$ time and $O\left(m \cdot S\left(\frac{n(1-\lambda)}{m} + \lambda n\right)\right)$ space, which is at the same order of magnitude comparing with the original costs for most graph-based ANNS algorithms.

## 4 Two-stage search on *CSPG*

The search process of *CSPG* is divided into two stages, i.e., *fast approaching* and *precise search*. Specifically, the first stage aims to quickly approach the query vector by using only one proximity graph, while the second stage will carefully search around by considering all partitions for the final closest vectors. Traditional beam search on a single proximity graph maintains a fixed-size candidate set. In contrast, *CSPG* uses different sizes $ef_1$ and $ef_2$ for the two stages respectively, where $ef_1 < ef_2$. Algorithm 1 outlines the procedure of searching on the *CSPG* index.

### 4.1 The First Stage: exploring single partition for fast approaching

In the first stage, the algorithm quickly approaches the query nearby with a shorter search length and fewer neighbor expansions. As shown in Algorithm 1, it conducts a greedy beam search discussed in Section 1 on the graph $\mathcal{G}_1$.

**Example 2.** *Given a dataset $\mathcal{D}$ and a query vector $\otimes$, We build CSPG and conduct the first stage search ($ef_1 = 1$) to approach the nearby region of the query within just 1 step (Figure 2).*

Each proximity graph in *CSPG($\mathcal{D}$)* is smaller and sparser than the proximity graph for the whole dataset $\mathcal{D}$ (denoted as *PG($\mathcal{D}$)*). This sparsity allows the first stage search to use larger steps and fewer moves to approximate the query. On the other hand, *CSPG* uses a smaller candidate size $ef_1$, eliminating some expansions that do not contribute to the final results.

### 4.2 The Second Stage: cross-partition expansion for precise search

After the greedy search in the first stage, the candidate set contains the closest vector delivered in the first stage from the first partition $\mathcal{P}_1$ w.r.t. the query vector. As shown in Algorithm 1, after resetting the visited set, it continues the greedy beam search with a size $ef_2$ for the candidate set $\mathcal{L}$. In the second stage, the significant difference from the first stage lies in line 13. Specifically, if the expanded neighbor $u$ is a routing vector, all its instances in all partitions will be pushed into $\mathcal{L}$. This approach allows the search process to dynamically traverse different proximity graphs, maximizing the search space to include as many potential closest vectors as possible.

**Example 3.** *This example continues the search from Example 2. For comparison, let us consider the traditional greedy beam search within $PG(\mathcal{D})$. As shown in Figure 1b, it takes 6 steps to approach the nearest neighbor $v_9$ (with a fixed candidate set size $ef = 3$). For CSPG, it first conducts the first stage, then switches the candidate set size from $ef_1 = 1$ to $ef_2 = 3$ and enters the second stage for a*

*more precise search in the query nearby as shown in Figure 2 (the two stages have 4 steps in total). In the second stage, CSPG expands the neighbors of $v_3^1$, $\mathcal{N}(v_3^1) = \{v_4^1\}$, in $\mathcal{G}_1$. Since $v_4$ is a routing vector, both $v_4^2$ and $v_4^1$ are added to $\mathcal{L}$. Next, the algorithm expands $v_4^2$, updating $\mathcal{L}$ to $\{v_7^1, v_7^2, v_4^1\}$ as $v_7$ is also a routing vector. Then, by expanding $v_7^1$, we reach the closest node $v_9^1$ in $\mathcal{G}_1$.*

Due to the sparsity of each proximity graph in *CSPG*, the search on *CSPG($\mathcal{D}$)* approaches the query results faster than *PG($\mathcal{D}$)*. Moreover, some expansions may be removed. For example, with a candidate size of $ef_2 = 2$, the candidate set in Step 2 would be $\{v_7^1, v_7^2\}$, removing $v_4^1$ due to the limited size, ignoring the unnecessary expansion to $v_5^1$.

In *PG($\mathcal{D}$)*, redundant vectors are mainly used to merge graphs. In contrast, *CSPG* leverages the distribution of routing vectors to ensure that the expansion of one graph aids in reducing expansions in other graphs. This means a position reached by one graph can be accessed by other graphs without additional expansion. For example, moving from $v_4^2$ to $v_7^2$ in $\mathcal{G}_2$ allows continuing the search from $v_7^1$ to $v_9^1$ in $\mathcal{G}_1$. The search across multiple graphs in the second stage appears as though it is conducted within a single proximity graph. The total number of steps to traverse the query nearby is 3, fewer than the 4 steps in *PG($\mathcal{D}$)* while maintaining the same precision. In practice, with appropriate partitioning, *CSPG* outperforms traditional *PG* algorithms, as discussed in Section 6.

## 5  Analysis of search efficiency

For most graph-based ANNS algorithms, calculating the distance between two vectors usually dominates the overall search time. In this section, we focus on the number of distance computations during query processing, denoted by $C$. We will show that the expected cost $\mathbb{E}[C]$ for the proposed *CSPG* method is lower than the traditional PG under certain constraints.

### 5.1  The expected number of distance computations

Following the setting of previous work [6], assume that the start vector $p$ and query vector $q$ are randomly selected from the $d$-dimensional vector dataset. On the proximity graph $\mathcal{G}$ built on the dataset, the greedy search sequence is denoted by $p \rightsquigarrow q = \{v_1, v_2, ..., v_t\}$, where $v_1 = p$, $v_t = q$. Denote $|p \rightsquigarrow q|$ as its length. The search sequence length of an MSNET equals its search path length, as the search consistently approaches the query without backtracking. Recall that when exploring each node, all its unvisited neighbors are pushed into the candidate set, and their distance to $q$ is computed. By assuming that the average degree of $\mathcal{G}$ is $\sigma$, the expected number of distance computation is $\mathbb{E}[C] = \sigma \mathbb{E}[|p \rightsquigarrow q|]$.

### 5.2  Expected search sequence length on Monotonic Search Network

As introduced in Section 2.3, for a Monotonic Search Network (MSNET), the expected length of the search sequence $\mathbb{E}^M[|p \rightsquigarrow u|] = \mathcal{O}\left(n^{\frac{1}{d}} \log n^{\frac{1}{d}} / \Delta r\right)$ [6]. In *CSPG* schema, assume that there are $m$ partitions ($m \ll n$), and the proximity graph $\mathcal{G}_i$ on each partition $\mathcal{P}_i$ is a MSNET. Since we randomly select routing vectors $RV$ and then randomly divide the other vectors into $m$ partitions, the distribution of vectors is the same in each partition. Then, we have the following theorem.

**Theorem 1.** *Given a start vector $s \in RV$ and a query vector $q \in RV$, by performing greedy beam search from $s$ on each MSNET $\mathcal{G}_i$ independently, we can obtain $m$ monotonic paths $s^i \rightsquigarrow q^i$, where $s^i, q^i \in \mathcal{G}_i$. It holds that $\mathbb{E}^{\mathcal{G}_i}[|s^i \rightsquigarrow q^i|] = \mathbb{E}^{\mathcal{G}_j}[|s^j \rightsquigarrow q^j|]$ for $1 \le i, j \le m$.*

*Proof.* Since the distribution of vectors in each graph $\mathcal{G}_i$ are the same, the assumptions for deriving the expected path length in [6] remain unchanged, thus they have the same expected path length.  □

By starting the search on a random entry vector $p$ in proximity graph $\mathcal{G}_1$, the routing vectors help us to travel across different partitions $\mathcal{P}_i$ in the second stage. Thus, the search sequence of *CSPG* is composed of several sub-sequences from different graphs $\mathcal{G}_i$. The following theorem reveals that the expected search sequence length of *CSPG* is the same as the case of searching on the *PG($\mathcal{G}_1$)*.

**Theorem 2.** *Denote $\mathbb{E}^{CSPG}[|p \rightsquigarrow q|]$ as the expected length of search sequence in CSPG. Denote $\mathbb{E}^{\mathcal{G}_i}[|p \rightsquigarrow q|]$ as the expected sequence length when searching only on the graph $\mathcal{G}_i$. It holds that*
$$\mathbb{E}^{CSPG}[|p \rightsquigarrow q|] = \mathbb{E}^{\mathcal{G}_i}[|p \rightsquigarrow q|].$$

Please refer to Appendix C for the detailed proofs. Based on Theorem 2, when the proximity graphs in *CSPG* are MSNET, the expected search path length is

$$\mathbb{E}^{CSPG}[|p \rightsquigarrow q|] = \mathbb{E}^{\mathcal{G}_i}[|p \rightsquigarrow q|] = \mathcal{O}\left(\left(\lambda n + \frac{n(1-\lambda)}{m}\right)^{\frac{1}{d}} \log \left(\lambda n + \frac{n(1-\lambda)}{m}\right)^{\frac{1}{d}} / \Delta r\right).$$

## 5.3 Approximate Monotonic Search Network

Most proximity graphs in practice are the approximation of MSNET, where the search path may have detours due to the lack of some necessary monotonic paths, resulting in distance backtracking. Given a query vector $q$, we say that vector $u$ conquers $q$ iff $\exists v \in \mathcal{N}(u), \delta(v, q) < \delta(u, q)$, denoted by $u \succ q$ ($u \neq q$). For a certain vector $u$ in the search path $p \rightsquigarrow q$, distance backtracking is avoided iff $u \succ q$, since when exploring $u$, we can visit $v$ to strictly decrease the distance w.r.t. $q$.

Intuitively, for the proximity graph $\mathcal{G}$, when the degree of every vector $u$ is large enough, $u \succ q$ is likely to be met for any query vector $q \in \mathcal{G}$, which help to avoid distance backtracking. However, since the average degree is usually limited in practical proximity graphs, there is a probability that vector $u$ conquers a random query vector $q \in \mathcal{G}$, formally defined as $\rho(u) = \frac{\sum_{q \in \mathcal{G}} \mathbb{I}(u \succ q)}{n}$,. where $\mathbb{I}(u \succ q) = 1$ iff $u \succ q$. Next, we introduce the *Approximate Monotonic Search Network (AMSNET)*, which reduces distance backtracking by maximizing $\rho(u)$ of each vector $u$.

**Definition 7** (Approximate Monotonic Search Network, shorted as AMSNET). *Given a dataset $\mathcal{D}$ of $n$ vectors, a proximity graph $\mathcal{G}$ built on $\mathcal{D}$ is called an approximate monotonic search network iff for every vector $u \in \mathcal{D}$, its neighbor set $\mathcal{N}(u)$ satisfies that $|\mathcal{N}(u)| \leq \sigma$ while maximizing $\rho(u)$.*

**Theorem 3.** *For datasets with the same distribution, as the number of vectors $n$ decreases, $\rho(u)$ is monotonically non-decreasing.*

Please refer to Appendix C for detailed proofs. Since AMSNET allows distance backtracking, there exists a detour factor $w > 1$ for the expected search sequence length. Specifically, when the underlying proximity graphs of *CSPG* are AMSNETs, the expected search sequence length is $\hat{\mathbb{E}}^{CSPG}[|p \rightsquigarrow q|] = \mathcal{O}\left(w \left(\lambda n + \frac{n(1-\lambda)}{m}\right)^{\frac{1}{d}} \log \left(\lambda n + \frac{n(1-\lambda)}{m}\right)^{\frac{1}{d}} / \Delta r\right)$. Moreover, according to Theorem 3, for every vector $u$, as dataset size $n$ decreases, $\rho(u)$ is non-decreasing. In other words, the probability of distance backtracking at every vector is non-increasing as $n$ decreases, thus $w$ is non-increasing as $n$ decreases. We confirm the monotonicity of $w$ in Section 6.4.

## 5.4 Speedup analysis for *CSPG*

For the dataset $\mathcal{D}$ of $n$ vectors, we have the following assumptions as discussed above: (1) $\exists \kappa$, $\kappa V_C \geq V_B(\cdot, R)$. (2) $m > 1$, $\lambda < 1$, and the vector distribution of each partition $\mathcal{P}_i$ is the same as $\mathcal{D}$. (3) The proximity graphs built in *PG* and *CSPG* are AMSNETs with a degree upper bound of $\sigma$.

When the proximity graph is AMSNET, the expected path length for *PG* method is $\hat{\mathbb{E}}^{PG}[|p \rightsquigarrow q|] = \mathcal{O}(n^{\frac{1}{d}} \log n^{\frac{1}{d}} / \Delta r^{PG})$, where $\Delta r^{PG}$ is the minimum distance difference for any non-isosceles triangle on $\mathcal{D}$. Similarly, $\Delta r^{CSPG}$ is defined for each partition $\mathcal{P}_i$ in *CSPG*. And the detour factor $w$ for *PG* and *CSPG* are denoted as $w^{PG}$ and $w^{CSPG}$, respectively. Thus, considering the expected number of distance computations, the speedup ratio of *CSPG* over *PG* is

$$Speedup = \frac{\sigma \times \hat{\mathbb{E}}^{PG}[|p \rightsquigarrow q|]}{\sigma \times \hat{\mathbb{E}}^{CSPG}[|p \rightsquigarrow q|]} = \left(\frac{w^{PG}}{w^{CSPG}} \times \frac{\Delta r^{CSPG}}{\Delta r^{PG}}\right) \times \frac{n^{\frac{1}{d}} \log n^{\frac{1}{d}}}{\left(\lambda n + \frac{n(1-\lambda)}{m}\right)^{\frac{1}{d}} \log \left(\lambda n + \frac{n(1-\lambda)}{m}\right)^{\frac{1}{d}}}$$

Define $\alpha = \frac{w^{PG}}{w^{CSPG}} \times \frac{\Delta r^{CSPG}}{\Delta r^{PG}}$. Since each proximity graph in *CSPG* are smaller than that in *PG*, and Section 5.3 shows that $w$ is non-increasing as $n$ decreases, $w^{PG} \geq w^{CSPG}$. Also, since $\Delta r$ decreases as $n$ increases [6], we have $\Delta r^{CSPG} \geq \Delta r^{PG}$. Thus, it holds that $\alpha \geq 1$. Next, we consider $\beta = \frac{n^{\frac{1}{d}} \log n^{\frac{1}{d}}}{\left(\lambda n + \frac{n(1-\lambda)}{m}\right)^{\frac{1}{d}} \log \left(\lambda n + \frac{n(1-\lambda)}{m}\right)^{\frac{1}{d}}}$. In *CSPG*, each partition has less than $n$ vectors when $m > 1$ and $\lambda < 1$, i.e., $\lambda n + \frac{n(1-\lambda)}{m} < n$ and $\beta > 1$. Thus, $Speedup = \alpha\beta > 1$, which ensures that *CSPG* always outperforms *PG* when using *AMSNET*. As the dataset size $n$ increases, $\beta$ is decreasing and we have $\lim_{n \to \infty} Speedup = \alpha \left(\frac{m}{(m-1)\lambda+1}\right)^{\frac{1}{d}}$. Moreover, when $n \to \infty$ and $d$ is increasing, $\beta$ is also decreasing and $\lim_{d \to \infty} Speedup = \alpha$.

## 6 Evaluation

### 6.1 Experimental setup

As summarized in Table 3, four benchmark datasets are used in our experiments, which are the most commonly used public datasets come from *Ann-benchmarks* [48].

Table 1: Comparison for index construction cost, in which DS is the data size (MB), IS is the graph index size (MB), and IT is the index construction time (s)

| index | | SIFT1M | | | GIST1M | | | DEEP1M | | | SIFT10M | | |
|---|---|---|---|---|---|---|---|---|---|---|---|---|---|
| | | DS | IS | IT | DS | IS | IT | DS | IS | IT | DS | IS | IT |
| PG | HNSW | 488 | 253 | 33 | 3,662 | 254 | 237 | 366 | 251 | 28 | 1,221 | 2,596 | 416 |
| | Vamana | | 126 | 97 | | 120 | 388 | | 128 | 90 | | 1296 | 1,122 |
| | HCNNG | | 44 | 85 | | 53 | 390 | | 54 | 79 | | 633 | 742 |
| CSPG | HNSW | | 389 | 50 | | 380 | 382 | | 387 | 48 | | 3,894 | 627 |
| | Vamana | | 195 | 145 | | 186 | 608 | | 192 | 136 | | 1,938 | 1,627 |
| | HCNNG | | 77 | 131 | | 81 | 465 | | 86 | 119 | | 934 | 1,128 |

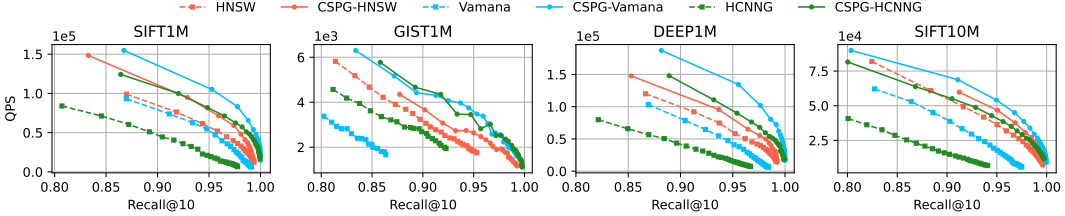

Figure 3: QPS v.s. recall curves for comparing query performance

Three well-known graph-based ANNS algorithms *HNSW* [2], *Vamana* [46], and *HCNNG* [45] are selected as baselines, which achieved competitive performance on previous studies [16, 49]. Also, we integrate each of them in the proposed *CSPG* method, resulting in three methods *CSPG-HNSW*, *CSPG-Vamana*, and *CSPG-HCNNG*. During the query phase, we set $k = 10$ and the quality of query results is evaluated by $recall@10$. The detailed index construction and query processing parameters are listed in Appendix D. The efficiency of an algorithm is measured by Queries Per Second (QPS), which is defined as the number of queries processed per second. And the answer quality is evaluated by $recall@k = \frac{|S \cap T|}{|T|}$, which measures the overlap between the retrieved results $S$ and the ground truth $T$. All experiments are conducted on a machine with Intel Xeon Gold 6136 CPU @3.00GHz and 128GB memory. We use 24 threads for both index construction and query processing. All our source codes are available at `https://github.com/PUITAR/CSPG`.

## 6.2 Evaluating query performance

Figure 3 presents the QPS-recall curve of the *CSPG* method built upon each graph-based ANNS algorithm, comparing it with the corresponding traditional *PG* implementation. For all these datasets and graph-based algorithms, *CSPG* method consistently improves the overall query performance. Specifically, *CSPG* helps the *Vamana* and *HCNNG* indices to achieve at least 1.5x speedup at a fixed recall of 0.9 on all the datasets. Such acceleration is due to the significantly reduced number of distance computations, as illustrated by Figure 10. The practical superior query performance of *CSPG* also aligns with the theoretical analysis in Section 5.

## 6.3 Evaluating index construction

By default, we use a sampling ratio of $\lambda = 0.5$ to divide the data into $m = 2$ partitions. Thus, the total number of vectors in the two proximity graphs is $(1 - \lambda)n + \lambda nm = 1.5n$. As shown in Table 1, for each graph-based ANNS algorithm, the index construction time and index size of *CSPG* is roughly 1.5x of the traditional *PG* version. Though *CSPG* has a larger index size, it is still affordable since the memory consumption of the vector raw data is much larger than the indices.

## 6.4 Impact of factors and parameters

In this section, we investigate how potential factors and parameters affect the query performance.

**Varying the dataset size**. Take dataset SIFT10M as an example, we randomly sample 0.1, 0.2, 0.5, 2, and 5 million vectors and build the graph-based ANNS indices and the corresponding *CSPG* method, using the default parameter settings. As shown in Figure 4, *CSPG* always achieves better performance compared with original algorithms over all the sampled datasets. For each graph-based ANNS algorithm, though *CSPG* always helps to improve the performance, such improvement becomes less obvious. It aligns with our theoretical results in Section 5.4, which shows that as $n$ increases, the speedup ratio decreases but converges to a value larger than 1.

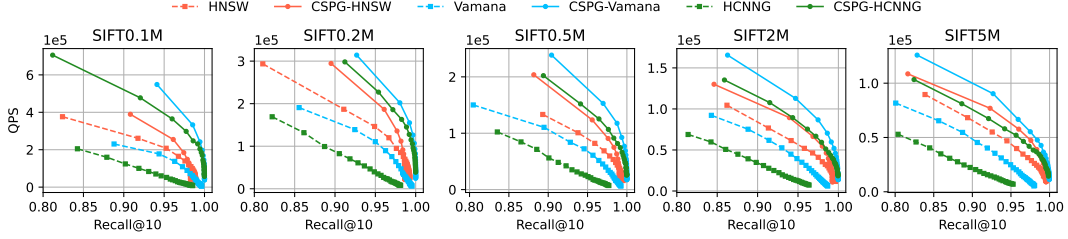

Figure 4: Query performance when varying the dataset size $n$

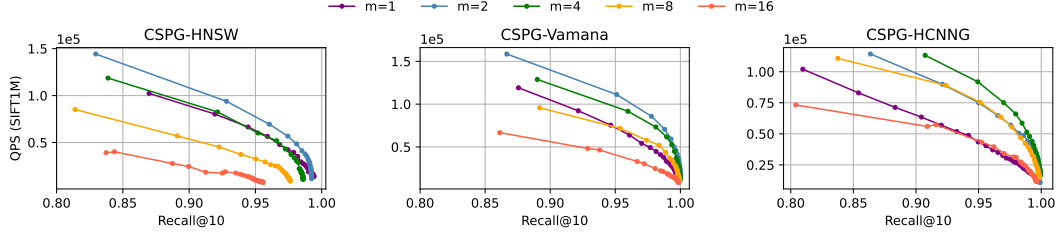

Figure 5: Query performance when varying the number of partitions $m$

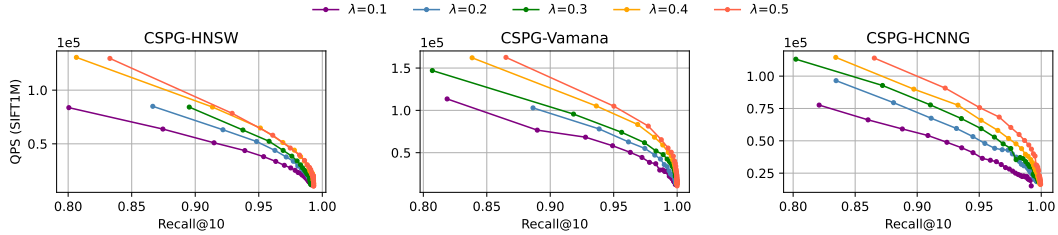

Figure 6: Query performance when varying the sampling ratio $\lambda$

**Varying the number of partitions.** Since the number of partitions $m$ affects the *CSPG* index quality, we conduct experiments for evaluating the query performance with $m = 1, 2, 4, 8, 16$ partitions, in which $m = 1$ indicates the original graph-based ANNS algorithm without partition. Note that the other parameters are the same as the default settings. Intuitively, larger $m$ results in fewer vectors in each partition, and *CSPG* seems to achieve better performance. However, large $m$ may decrease the similarity of vector distribution among different partitions, which contradicts the assumptions of the same vector distribution discussed in Section 5.4. Therefore, for different datasets with various distributions, choosing an appropriate parameter $m$ is crucial. As shown in Figure 5, for SIFT1M dataset, dividing all vectors into $2$ or $4$ partition results in better QPS-recall curves. Figure 11 shows that the optimal value of $m$ for the other datasets ranges from $2$ to $8$.

**Varying the sampling ratio.** When constructing the *CSPG* schema, the sampling ratio $\lambda$ is used to randomly sample $\lambda n$ routing vectors before dataset partition. By using the default values of all the other parameters, Figure 6 reveals that for SIFT1M dataset, larger $\lambda$ tends to improve the query performance of *CSPG*. And it also holds for other datasets, as shown in Figure 12. This is because more routing vectors help to navigate the search traveling across different partitions efficiently. Also, more routing vectors result in more shared vectors in each partition, increasing the similarity of vector distribution among different partitions, which is more aligned with the assumptions of the same vector distribution discussed in Section 5.4.

**Varying the candidate set sizes.** *CSPG* has two parameters $ef_1$ and $ef_2$ for searching, limiting the size of the candidate set in the two stages, respectively. As shown in Figure 7 and Figure 13, we try different $ef_1 = 1, 2, 4, 8, 16$ and obtain 5 QPS v.s. recall curves. The marks in each curve are obtained by varying the value of $ef_2$ uniformly picked from $[10, 300]$. In most cases, $ef_1 = 1$ provides the best query performance, which aligns with our goal of the first stage fast approaching.

**Statistics for detour factor.** We randomly sample $0.1$, $0.2$, $0.5$, $2$, and $5$ million vectors from SIFT10M dataset. By using *CSPG* with the default parameter settings, at different $Recall@k$, we obtain the empirical detour factor $w = \frac{\text{length of search sequence}}{\text{length of search sequence} - \text{number of distance backtracking}}$ averaged for all

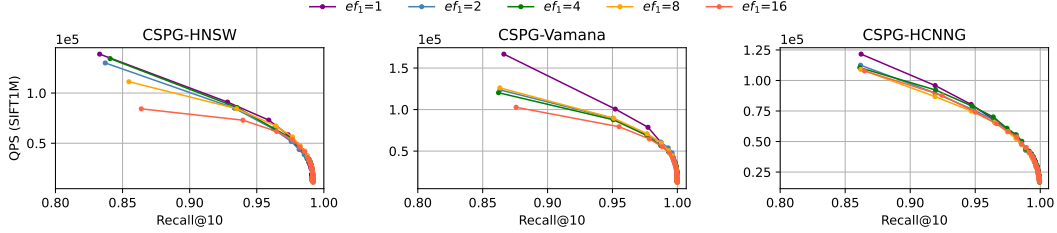

Figure 7: Query performance when varying the candidate set size $ef_1$ in the first stage

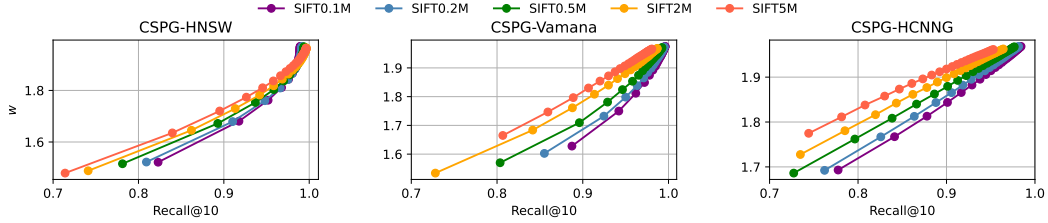

Figure 8: Detour factor when varying the dataset size $n$

search paths. As shown in Figure 8, larger $n$ results in larger $w$ at a fixed $Recall@k$, which aligns with our discussion in Section 5.3 that $w$ is non-increasing as $n$ decreases.

### 6.5 Hyperparameter grid search and evaluation with ANN-Benchmarks

To show the Pareto frontiers with the optimal hyperparameters, we conduct a grid search experiment to identify the most effective hyperparameter combinations for each baseline on SIFT1M dataset. As presented in Appendix D, we summarise the parameter selection sets for different indices in Table 4, and the results of the grid search experiment are illustrated in Figure 14.

Next, we conduct a more comprehensive evaluation using the ANN-Benchmarks [48], which is a widely used benchmarking environment. For each algorithm, we choose the optimal parameters from the grid search experiment with the highest QPS when Recall@10 $= 0.9 \pm 5e^{-3}$ on SIFT1M dataset. As shown in Figure 9, enhanced with the proposed *CSPG* framework, representative algorithms such as *HCNNG*, *HNSW*, *Vamana* and *NSG* can achieve better performance on SIFT1M dataset.

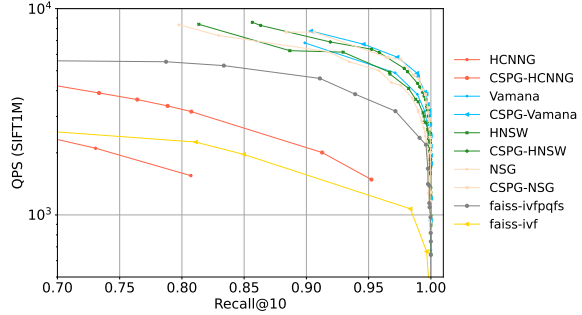

Figure 9: QPS v.s. Recall@10 curve on SIFT1M with the optimal parameters in ANN-Benchmarks

## 7 Conclusion

We proposed a novel graph-based indexing schema named *CSPG* for Approximate Nearest Neighbor Search (ANNS), which is compatible with the current leading graph-based approaches in high-recall scenarios. Furthermore, we propose a novel search algorithm for the *CSPG* schema, which uses a two-stage strategy and a cross-partition expansion to reduce meaningless expansion during the graph search and make the process more focused on the parts related to the answer. Next, we analyze the expectation of *CSPG*'s search amount, establish a speedup model, and prove that *CSPG* can always have an advantage. Finally, we investigate the advantages of *CSPG* through experiments and carry out a more detailed evaluation of the key factors affecting the performance of *CSPG*.

## Acknowledgments and Disclosure of Funding

This work was substantially supported Key Projects of the National Natural Science Foundation of China (Grant No. U23A20496) and Shanghai Science and Technology Innovation Action Plan (Grant No. 21511100401). Weiguo Zheng is the corresponding author.

## References

[1] Akhil Arora, Sakshi Sinha, Piyush Kumar, and Arnab Bhattacharya. Hd-index: Pushing the scalability-accuracy boundary for approximate knn search in high-dimensional spaces. *arXiv preprint arXiv:1804.06829*, 2018.

[2] Yu A Malkov and Dmitry A Yashunin. Efficient and robust approximate nearest neighbor search using hierarchical navigable small world graphs. *IEEE transactions on pattern analysis and machine intelligence*, 42(4):824–836, 2018.

[3] Kazuo Aoyama, Kazumi Saito, Hiroshi Sawada, and Naonori Ueda. Fast approximate similarity search based on degree-reduced neighborhood graphs. In *Proceedings of the 17th ACM SIGKDD international conference on Knowledge discovery and data mining*, pages 1055–1063, 2011.

[4] Minjia Zhang and Yuxiong He. Zoom: Ssd-based vector search for optimizing accuracy, latency and memory. *arXiv preprint arXiv:1809.04067*, 2018.

[5] Rentong Guo, Xiaofan Luan, Long Xiang, Xiao Yan, Xiaomeng Yi, Jigao Luo, Qianya Cheng, Weizhi Xu, Jiarui Luo, Frank Liu, et al. Manu: a cloud native vector database management system. *Proceedings of the VLDB Endowment*, 15(12):3548–3561, 2022.

[6] Cong Fu, Chao Xiang, Changxu Wang, and Deng Cai. Fast approximate nearest neighbor search with the navigating spreading-out graph. *arXiv preprint arXiv:1707.00143*, 2017.

[7] Wenhui Zhou, Chunfeng Yuan, Rong Gu, and Yihua Huang. Large scale nearest neighbors search based on neighborhood graph. In *2013 International Conference on Advanced Cloud and Big Data*, pages 181–186. IEEE, 2013.

[8] Chun Jiang Zhu, Tan Zhu, Haining Li, Jinbo Bi, and Minghu Song. Accelerating large-scale molecular similarity search through exploiting high performance computing. In *2019 IEEE International Conference on Bioinformatics and Biomedicine (BIBM)*, pages 330–333. IEEE, 2019.

[9] Myron Flickner, Harpreet Sawhney, Wayne Niblack, Jonathan Ashley, Qian Huang, Byron Dom, Monika Gorkani, Jim Hafner, Denis Lee, Dragutin Petkovic, et al. Query by image and video content: The qbic system. *computer*, 28(9):23–32, 1995.

[10] Atsutake Kosuge and Takashi Oshima. An object-pose estimation acceleration technique for picking robot applications by using graph-reusing k-nn search. In *2019 First International Conference on Graph Computing (GC)*, pages 68–74. IEEE, 2019.

[11] Thomas Cover and Peter Hart. Nearest neighbor pattern classification. *IEEE transactions on information theory*, 13(1):21–27, 1967.

[12] Yitong Meng, Xinyan Dai, Xiao Yan, James Cheng, Weiwen Liu, Jun Guo, Benben Liao, and Guangyong Chen. Pmd: An optimal transportation-based user distance for recommender systems. In *Advances in Information Retrieval: 42nd European Conference on IR Research, ECIR 2020, Lisbon, Portugal, April 14–17, 2020, Proceedings, Part II 42*, pages 272–280. Springer, 2020.

[13] Badrul Sarwar, George Karypis, Joseph Konstan, and John Riedl. Item-based collaborative filtering recommendation algorithms. In *Proceedings of the 10th international conference on World Wide Web*, pages 285–295, 2001.

[14] Tom Brown, Benjamin Mann, Nick Ryder, Melanie Subbiah, Jared D Kaplan, Prafulla Dhariwal, Arvind Neelakantan, Pranav Shyam, Girish Sastry, Amanda Askell, et al. Language models are few-shot learners. *Advances in neural information processing systems*, 33:1877–1901, 2020.

[15] Yunfan Gao, Yun Xiong, Xinyu Gao, Kangxiang Jia, Jinliu Pan, Yuxi Bi, Yi Dai, Jiawei Sun, and Haofen Wang. Retrieval-augmented generation for large language models: A survey. *arXiv preprint arXiv:2312.10997*, 2023.

[16] Mengzhao Wang, Xiaoliang Xu, Qiang Yue, and Yuxiang Wang. A comprehensive survey and experimental comparison of graph-based approximate nearest neighbor search. *arXiv preprint arXiv:2101.12631*, 2021.

[17] Wen Li, Ying Zhang, Yifang Sun, Wei Wang, Mingjie Li, Wenjie Zhang, and Xuemin Lin. Approximate nearest neighbor search on high dimensional data—experiments, analyses, and improvement. *IEEE Transactions on Knowledge and Data Engineering*, 32(8):1475–1488, 2019.

[18] Keinosuke Fukunaga and Patrenahalli M. Narendra. A branch and bound algorithm for computing k-nearest neighbors. *IEEE transactions on computers*, 100(7):750–753, 1975.

[19] Norbert Beckmann, Hans-Peter Kriegel, Ralf Schneider, and Bernhard Seeger. The r*-tree: An efficient and robust access method for points and rectangles. In *Proceedings of the 1990 ACM SIGMOD international conference on Management of data*, pages 322–331, 1990.

[20] Chanop Silpa-Anan and Richard Hartley. Optimised kd-trees for fast image descriptor matching. In *2008 IEEE Conference on Computer Vision and Pattern Recognition*, pages 1–8. IEEE, 2008.

[21] Jon Louis Bentley. Multidimensional binary search trees used for associative searching. *Communications of the ACM*, 18(9):509–517, 1975.

[22] Hosagrahar V Jagadish, Beng Chin Ooi, Kian-Lee Tan, Cui Yu, and Rui Zhang. idistance: An adaptive b+-tree based indexing method for nearest neighbor search. *ACM Transactions on Database Systems (TODS)*, 30(2):364–397, 2005.

[23] Qiang Huang, Jianlin Feng, Yikai Zhang, Qiong Fang, and Wilfred Ng. Query-aware locality-sensitive hashing for approximate nearest neighbor search. *Proceedings of the VLDB Endowment*, 9(1):1–12, 2015.

[24] Aristides Gionis, Piotr Indyk, Rajeev Motwani, et al. Similarity search in high dimensions via hashing. In *Vldb*, pages 518–529, 1999.

[25] Jinyang Gao, Hosagrahar Visvesvaraya Jagadish, Wei Lu, and Beng Chin Ooi. Dsh: data sensitive hashing for high-dimensional k-nnsearch. In *Proceedings of the 2014 ACM SIGMOD international conference on Management of data*, pages 1127–1138, 2014.

[26] Yingfan Liu, Jiangtao Cui, Zi Huang, Hui Li, and Heng Tao Shen. Sk-lsh: an efficient index structure for approximate nearest neighbor search. *Proceedings of the VLDB Endowment*, 7(9):745–756, 2014.

[27] Yair Weiss, Antonio Torralba, and Rob Fergus. Spectral hashing. *Advances in neural information processing systems*, 21, 2008.

[28] Fabien André, Anne-Marie Kermarrec, and Nicolas Le Scouarnec. Cache locality is not enough: High-performance nearest neighbor search with product quantization fast scan. In *42nd International Conference on Very Large Data Bases*, page 12, 2016.

[29] Tiezheng Ge, Kaiming He, Qifa Ke, and Jian Sun. Optimized product quantization for approximate nearest neighbor search. In *Proceedings of the IEEE conference on computer vision and pattern recognition*, pages 2946–2953, 2013.

[30] Herve Jegou, Matthijs Douze, and Cordelia Schmid. Product quantization for nearest neighbor search. *IEEE transactions on pattern analysis and machine intelligence*, 33(1):117–128, 2010.

[31] Jianyang Gao and Cheng Long. Rabitq: Quantizing high-dimensional vectors with a theoretical error bound for approximate nearest neighbor search. *Proceedings of the ACM on Management of Data*, 2(3):1–27, 2024.

[32] Yury Malkov, Alexander Ponomarenko, Andrey Logvinov, and Vladimir Krylov. Approximate nearest neighbor algorithm based on navigable small world graphs. *Information Systems*, 45: 61–68, 2014.

[33] Sunil Arya and David M Mount. Approximate nearest neighbor queries in fixed dimensions. In *SODA*, volume 93, pages 271–280. Citeseer, 1993.

[34] Yubao Wu, Ruoming Jin, and Xiang Zhang. Fast and unified local search for random walk based k-nearest-neighbor query in large graphs. In *Proceedings of the 2014 ACM SIGMOD international conference on Management of Data*, pages 1139–1150, 2014.

[35] Kiana Hajebi, Yasin Abbasi-Yadkori, Hossein Shahbazi, and Hong Zhang. Fast approximate nearest-neighbor search with k-nearest neighbor graph. In *Twenty-Second International Joint Conference on Artificial Intelligence*, 2011.

[36] Zeyu Wang, Haoran Xiong, Zhenying He, Peng Wang, et al. Distance comparison operators for approximate nearest neighbor search: Exploration and benchmark. *arXiv preprint arXiv:2403.13491*, 2024.

[37] Jianyang Gao and Cheng Long. High-dimensional approximate nearest neighbor search: with reliable and efficient distance comparison operations. *Proceedings of the ACM on Management of Data*, 1(2):1–27, 2023.

[38] DW Dearholt, N Gonzales, and G Kurup. Monotonic search networks for computer vision databases. In *Twenty-Second Asilomar Conference on Signals, Systems and Computers*, volume 2, pages 548–553. IEEE, 1988.

[39] Godfried T Toussaint. The relative neighbourhood graph of a finite planar set. *Pattern recognition*, 12(4):261–268, 1980.

[40] Yun Peng, Byron Choi, Tsz Nam Chan, Jianye Yang, and Jianliang Xu. Efficient approximate nearest neighbor search in multi-dimensional databases. *Proceedings of the ACM on Management of Data*, 1(1):1–27, 2023.

[41] Franz Aurenhammer. Voronoi diagrams—a survey of a fundamental geometric data structure. *ACM Computing Surveys (CSUR)*, 23(3):345–405, 1991.

[42] Ben Harwood and Tom Drummond. Fanng: Fast approximate nearest neighbour graphs. In *Proceedings of the IEEE Conference on Computer Vision and Pattern Recognition*, pages 5713–5722, 2016.

[43] Jerzy W Jaromczyk and Godfried T Toussaint. Relative neighborhood graphs and their relatives. *Proceedings of the IEEE*, 80(9):1502–1517, 1992.

[44] Wei Dong, Charikar Moses, and Kai Li. Efficient k-nearest neighbor graph construction for generic similarity measures. In *Proceedings of the 20th international conference on World wide web*, pages 577–586, 2011.

[45] Javier Vargas Munoz, Marcos A Gonçalves, Zanoni Dias, and Ricardo da S Torres. Hierarchical clustering-based graphs for large scale approximate nearest neighbor search. *Pattern Recognition*, 96:106970, 2019.

[46] Suhas Jayaram Subramanya, Fnu Devvrit, Harsha Vardhan Simhadri, Ravishankar Krishnawamy, and Rohan Kadekodi. Diskann: Fast accurate billion-point nearest neighbor search on a single node. *Advances in Neural Information Processing Systems*, 32, 2019.

[47] Jing Wang, Jingdong Wang, Gang Zeng, Zhuowen Tu, Rui Gan, and Shipeng Li. Scalable k-nn graph construction for visual descriptors. In *2012 IEEE Conference on Computer Vision and Pattern Recognition*, pages 1106–1113. IEEE, 2012.

[48] Martin Aumüller, Erik Bernhardsson, and Alexander Faithfull. ANN-Benchmarks: A benchmarking tool for approximate nearest neighbor algorithms. *Information Systems*, 87:101374, 2020.

[49] Magdalen Dobson, Zheqi Shen, Guy E Blelloch, Laxman Dhulipala, Yan Gu, Harsha Vardhan Simhadri, and Yihan Sun. Scaling graph-based anns algorithms to billion-size datasets: A comparative analysis. *arXiv preprint arXiv:2305.04359*, 2023.

# A    Related graph-based ANNS algorithms

As discussed above, graph-based ANNS algorithms conduct a best-first greedy beam search on the proximity graphs to approach the closest nodes for a query vector. Their built proximity graphs can be classified into four categories [16, 40] as follows.

**Delaunay Graph (DG) [41, 42]**. It ensures that for any edge, no other vectors will be situated within the hypersphere defined by an edge connecting two vectors, where the hypersphere is centered at the midpoint of the edge and the length of the edge is the diameter. But when the dimension $d$ is large, DG tends to become a complete graph [42], rapidly increasing the costs of greedy search.

**Relative Neighborhood Graph (RNG)** [39]. It guarantees that for any edge between $p$ and $q$, no other vectors will reside within the $lune(p,q) = \{u \in \mathbb{R}^d \,|\, \delta(u,p) \leq \delta(p,q) \wedge \delta(u,q) \leq \delta(p,q)\}$. Compared to DG, RNG imposes stricter restrictions on its edges, thus decreasing the average degree [43]. Various works are based on RNG, including the Monotonic Relative Neighbor Graph (MRNG) [6] and FANNG [42].

**K-Nearest Neighbor Graph (KNNG)** [44]. In KNNG, neighbors of each vector $v \in \mathcal{D}$ are its top-$k$ nearest neighbors in $\mathcal{D}$, avoiding too many explored neighbors during searching. Since each node in KNNG has only $k$ neighbors, it cannot achieve the same level of connectivity as DG. NN-Descent [44] proposes a method for constructing KNNG. For constructing KNNG, NN-Descent [44] was proposed, which starts from a random graph and iteratively refines the graph towards the KNNG.

**Minimum Spanning Tree (MST) Graph** [45]. The MST utilizes distances between vectors as edge weights. Then, it performs hierarchical clustering on the dataset multiple times randomly, each time adding some edges to the edge set. MST can establish global connectivity with a minimal number of edges but may result in detours during searching.

Most well-known graph-based ANNS algorithms, including *HNSW* [2], *Vamana* [46], *HCNNG* [45], and *NSG* [6], do not exactly fit into one of the four categories. Instead, they usually evolve from one or more of these proximity graphs. For example, *Vamana* evolved from the combination of two prior algorithms. *HNSW* originated from both the DG and RNG algorithms, while *NSG* originated solely from the RNG algorithm.

# B    Algorithms and illustrations

---

**Algorithm 2** *CSPG* index construction

---

**Require:** dataset $\mathcal{D} = \{v_1, v_2, ..., v_n\}$, number of partitions $m$, and sampling ratio $\lambda \in [0, 1]$
**Ensure:** *CSPG* index $\mathcal{G} = \{\mathcal{G}_1, \mathcal{G}_2, ..., \mathcal{G}_m\}$, routing vectors $RV$
 1: $\mathcal{P}_i \leftarrow \emptyset$ for $1 \leq i \leq m$, $RV \leftarrow \emptyset$
 2: $RV \leftarrow$ random $\lfloor n \times \lambda \rfloor$ samples from $\mathcal{D}$
 3: **for** each vector $v$ in $\mathcal{D} \setminus RV$ **do**
 4:      add $v$ to a random sampled partition $\mathcal{P}_i$
 5: $\mathcal{G} \leftarrow$ construct a proximity graph $\mathcal{G}_i$ for each partition $\mathcal{P}_i$
 6: **return** $\mathcal{G}$ and $RV$

---

**Algorithm 3** *CSPG* vector insertion

---

**Require:** *CSPG* index $\mathcal{G} = \{\mathcal{G}_1, \mathcal{G}_2, ..., \mathcal{G}_m\}$, routing vectors $RV$, and vector $x$
 1: assign a random $i$, insert $x$ to $\mathcal{G}_i$ with the insertion method of the underlying graph index
 2: **if** $x$ is selected as route vector **then**
 3:      insert $x$ into $RV$
 4:      **for** each $j \in \{1, 2, ..., m\}$ s.t. $i \neq j$ **do**
 5:           insert $x$ to $\mathcal{G}_j$ with the insertion method of the underlying graph index

---

**Algorithm 4** *CSPG* vector deletion

---

**Require:** *CSPG* index $\mathcal{G} = \{\mathcal{G}_1, \mathcal{G}_2, ..., \mathcal{G}_m\}$, routing vectors $RV$, and vector $x$
 1: **for** for each $i \in \{1, 2, ..., m\}$ s.t. $x \in \mathcal{G}_i$ **do**
 2:      remove $x$ from $\mathcal{G}_i$ with the insertion method of the underlying graph index
 3: **if** $x$ is a route vector **then**
 4:      remove $x$ from $RV$

---

Table 2: Candidate Set for the search in the query nearby (dashed parallelogram).

| Step | *PG*'s Candidate Set | *CSPG*'s Candidate Set |
|------|---------------------|------------------------|
| 0 | $\{v_3\}$ | $\{v_3^1\}$ |
| 1 | $\{v_4\}$ | $\{v_4^2, v_4^1\}$ |
| 2 | $\{v_6, v_5\}$ | $\{v_7^1, v_7^2, v_4^1\}$ |
| 3 | $\{v_7, v_5\}$ | $\{\boldsymbol{v_9^1}, v_7^2, v_4^1\}$ |
| 4 | $\{\boldsymbol{v_9}, v_8, v_5\}$ | |

## C   Proofs for Section 5

**Theorem 2.** Denote $\mathbb{E}^{CSPG}[|p \rightsquigarrow q|]$ as the expected length of search sequence in *CSPG*. Denote $\mathbb{E}^{\mathcal{G}_i}[|p \rightsquigarrow q|]$ as the expected sequence length when searching only on the graph $\mathcal{G}_i$. It holds that

$$\mathbb{E}^{CSPG}[|p \rightsquigarrow q|] = \mathbb{E}^{\mathcal{G}_i}[|p \rightsquigarrow q|].$$

*Proof.* For the greedy search in *CSPG* starting from $\mathcal{G}_i$, denote $\vartheta \in RV$ as the first routing vector that the search sequence visits. Thus, $\Pr(\vartheta \text{ not exists}) + \sum_{s \in RV} \Pr(\vartheta = s) = 1$, where $\vartheta$ not exists indicates that the sequence never visits a routing vector before reaching the query vector $q$. Then, we have

$$\mathbb{E}^{CSPG}[|p \rightsquigarrow q|] = \Pr(\vartheta \text{ not exists})\mathbb{E}^{\mathcal{G}_i}\left[|p \rightsquigarrow q|\,\Big|\,\vartheta \text{ not exists}\right] +$$

$$\sum_{s \in RV} \Pr(\vartheta = s)\mathbb{E}^{CSPG}\left[|p \rightsquigarrow q|\,\Big|\,\vartheta = s\right]$$

$$= \Pr(\vartheta \text{ not exists})\mathbb{E}^{\mathcal{G}_i}\left[|p \rightsquigarrow q|\,\Big|\,\vartheta \text{ not exists}\right] +$$

$$\sum_{s \in RV} \Pr(\vartheta = s)\left(\mathbb{E}^{\mathcal{G}_i}[|p \rightsquigarrow s|] + \mathbb{E}^{CSPG}[|s \rightsquigarrow q|]\right).$$

Note that the term of $\mathbb{E}^{CSPG}[|s \rightsquigarrow q|]$ can be further expanded similarly. Since any search sequence is bounded by length $n$, such expansion can be done iteratively in a finite number. Consider the last expansion of

$$\mathbb{E}^{CSPG}[|s' \rightsquigarrow q|] = \Pr(\vartheta \text{ not exists})\mathbb{E}^{\mathcal{G}_i}\left[|s' \rightsquigarrow q|\,\Big|\,\vartheta \text{ not exists}\right] +$$

$$\sum_{s'' \in RV} \Pr(\vartheta = s'')\left(\mathbb{E}^{\mathcal{G}_i}[|s' \rightsquigarrow s''|] + \mathbb{E}^{CSPG}[|s'' \rightsquigarrow q|]\right).$$

The search path $s'' \rightsquigarrow q$ lies in one of the graphs. According to Theorem 1, since the path $s \rightsquigarrow q$ has the same expected length in any graph, it holds that $\mathbb{E}^{CSPG}[|s'' \rightsquigarrow q|] = \mathbb{E}^{\mathcal{G}_i}[|s'' \rightsquigarrow q|]$. Thus,

$$\mathbb{E}^{CSPG}[|s' \rightsquigarrow q|] = \Pr(\vartheta \text{ not exists})\mathbb{E}^{\mathcal{G}_i}\left[|s' \rightsquigarrow q|\,\Big|\,\vartheta \text{ not exists}\right] +$$

$$\sum_{s'' \in RV} \Pr(\vartheta = s'')\left(\mathbb{E}^{\mathcal{G}_i}[|s' \rightsquigarrow s''|] + \mathbb{E}^{\mathcal{G}_i}[|s'' \rightsquigarrow q|]\right)$$

$$= \Pr(\vartheta \text{ not exists})\mathbb{E}^{\mathcal{G}_i}\left[|s' \rightsquigarrow q|\,\Big|\,\vartheta \text{ not exists}\right] +$$

$$\sum_{s'' \in RV} \Pr(\vartheta = s'')\mathbb{E}^{\mathcal{G}_i}\left[|s' \rightsquigarrow q|\,\Big|\,\vartheta = s''\right]$$

$$= \mathbb{E}^{\mathcal{G}_i}[|s' \rightsquigarrow q|].$$

By recursively apply such induction, we can prove that $\mathbb{E}^{CSPG}[|p \rightsquigarrow q|] = \mathbb{E}^{\mathcal{G}_i}[|p \rightsquigarrow q|]$.   $\square$

**Theorem 3.** For datasets with the same distribution, as the number of vectors $n$ decreases, $\rho(u)$ is monotonically non-decreasing.

*Proof.* Given an AMSNET $\mathcal{G}$ build on a dataset $\mathcal{D}$ with $n$ vectors, for each vector $u \in \mathcal{G}$, $\rho(u)$ is the probability that $u$ conquers a random vector in $\mathcal{G}$. Consider the case that we randomly remove a vector $q^*$ from $\mathcal{D}$. For the AMSNET $\mathcal{G}'$ built on dataset $\mathcal{D}\backslash\{q^*\}$, denote $\rho'(u)$ as the probability that $u$ conquers a random vector in $\mathcal{G}'$. Next, we show that $\rho'(u) \geq \rho(u)$.

In AMSNET $\mathcal{G}$, since $q^*$ is randomly sampled from $\mathcal{D}$, the probability that $u \succ q^*$ is $\rho(u)$. Consider the following two cases.

- If $u \succ q^*$ holds in $\mathcal{G}$, $\sum_{q \in \mathcal{G}'} \mathbb{I}(u \succ q) \geq \sum_{q \in \mathcal{G}} \mathbb{I}(u \succ q) - 1$. It is because $\mathcal{G}'$ does not have $q^*$, and for building the AMSNET $\mathcal{G}'$ we can adjust the neighbors of $u$ and $u$ may conquer more vectors in $\mathcal{G}'$.

- If $u \succ q^*$ does not hold in $\mathcal{G}$, $\sum_{q \in \mathcal{G}'} \mathbb{I}(u \succ q) = \sum_{q \in \mathcal{G}} \mathbb{I}(u \succ q)$, since we can use the same neighbors of $u$ for both $\mathcal{G}$ and $\mathcal{G}'$.

Therefore, we have

$$
\begin{aligned}
\rho'(u) =& \frac{\rho(u)\left[\sum_{q \in \mathcal{G}'} \mathbb{I}(u \succ q)\right] + (1 - \rho(u))\left[\sum_{q \in \mathcal{G}'} \mathbb{I}(u \succ q)\right]}{n-1} \\
\geq& \frac{\rho(u)\left[\sum_{q \in \mathcal{G}} \mathbb{I}(u \succ q) - 1\right] + (1 - \rho(u))\left[\sum_{q \in \mathcal{G}} \mathbb{I}(u \succ q)\right]}{n-1} \\
=& \frac{\sum_{q \in \mathcal{G}} \mathbb{I}(u \succ q) - \rho(u)}{n-1} = \frac{n\rho(u) - \rho(u)}{n-1} = \rho(u).
\end{aligned}
$$

$\square$

## D  Experiments settings and results

The detailed index construction parameters are listed as follows, which are from previous studies [49] and *Ann-benchmarks* [48].

- *HNSW*. The degree upper bound $M = 32$, and the $efConstruction = 128$.

- *Vamana*. The degree upper bound $R = 32$, beam size $L = 128$, and pruning parameter $\alpha = 1.2$.

- *HCNNG*. The number of cluster trees $T = 10$, the leaf size of *MST* $Ls = 1000$, and the *MST* degree $s = 5$.

- *CSPG*. For fairness, for each graph-based algorithm used in *CSPG*, we slightly adjust its parameters to ensure the average degree of the built graphs is the same as the original algorithm.

The parameters $ef$ for each baseline algorithm and the $ef_2$ for *CSPG* method are uniformly picked from $[10, 300]$ to obtain *QPS* at different *Recall*. For *CSPG* method, we set parameter $ef_1 = 1$ and sampling ratio $\lambda = 0.5$ by default, while their impacts on performance are discussed in Section 6.4.

Table 3: Statistics of Datasets

| Dataset | Dimension | Data type | # Base | # Query |
|---------|-----------|-----------|--------|---------|
| SIFT1M | 128 | float | 1,000,000 | 10,000 |
| GIST1M | 960 | float | 1,000,000 | 1,000 |
| DEEP1M | 96 | float | 1,000,000 | 10,000 |
| SIFT10M | 128 | uint8 | 10,000,000 | 10,000 |

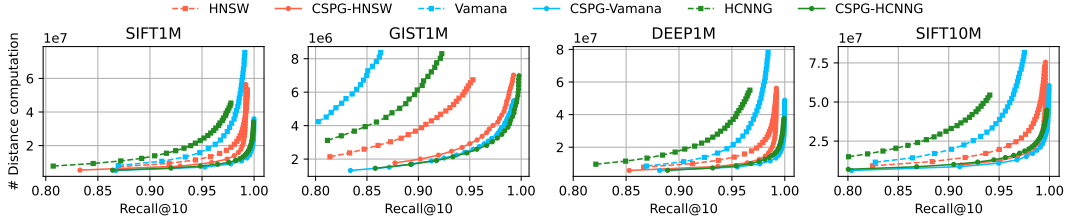

Figure 10: Number of distance computation v.s. recall curves for comparing query performance

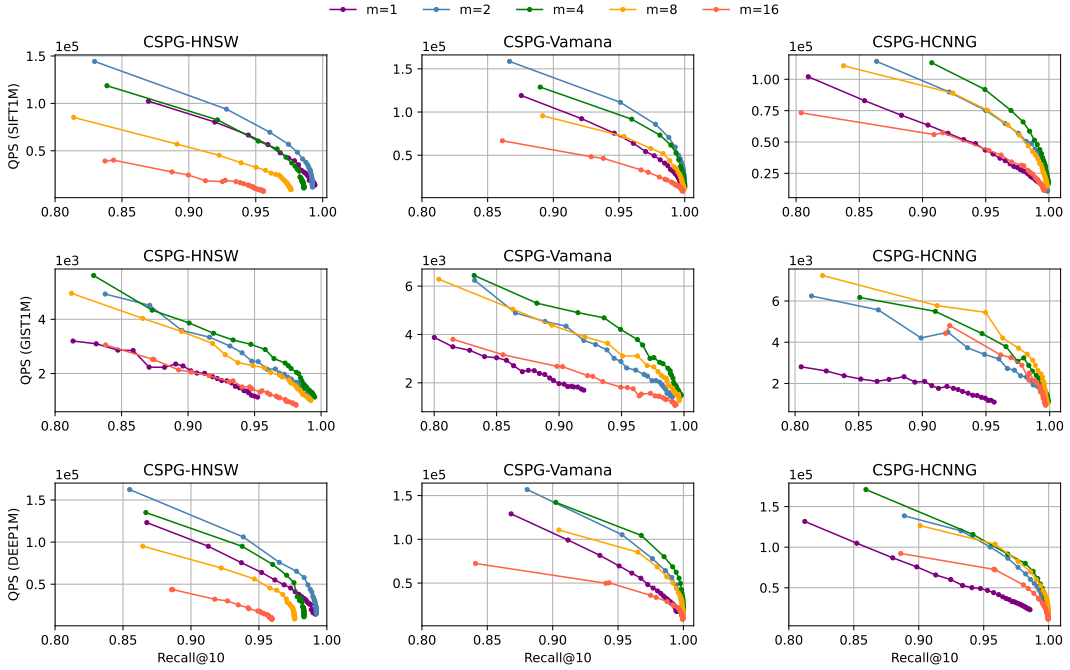

Figure 11: Query performance when varying the number of partitions $m$

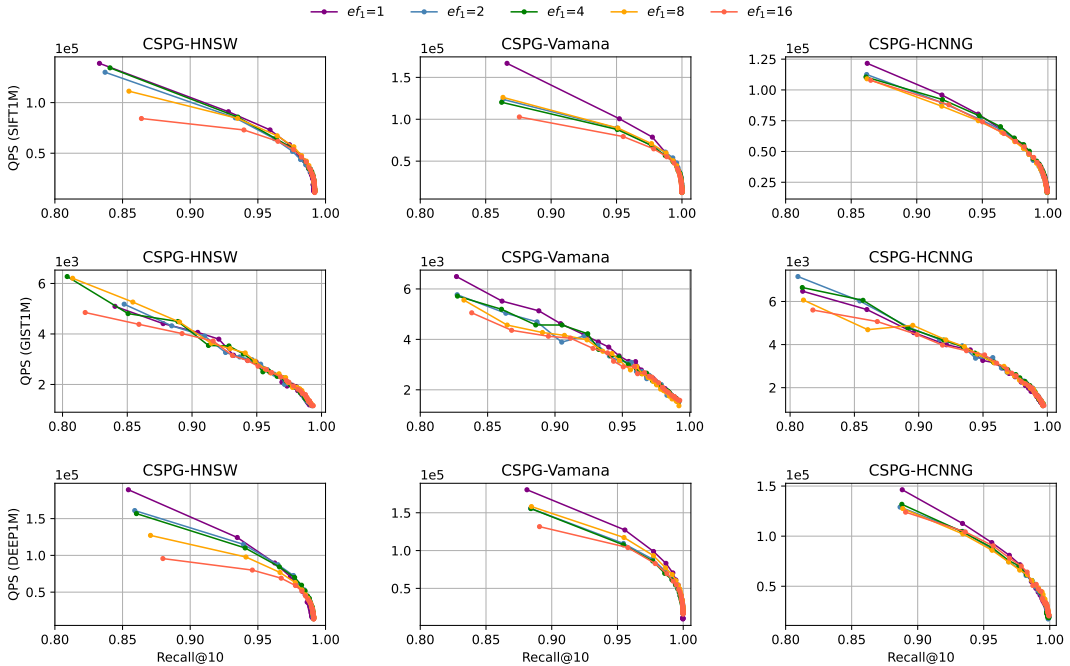

Figure 13: Query performance when varying the candidate set size $ef_1$ in the first stage

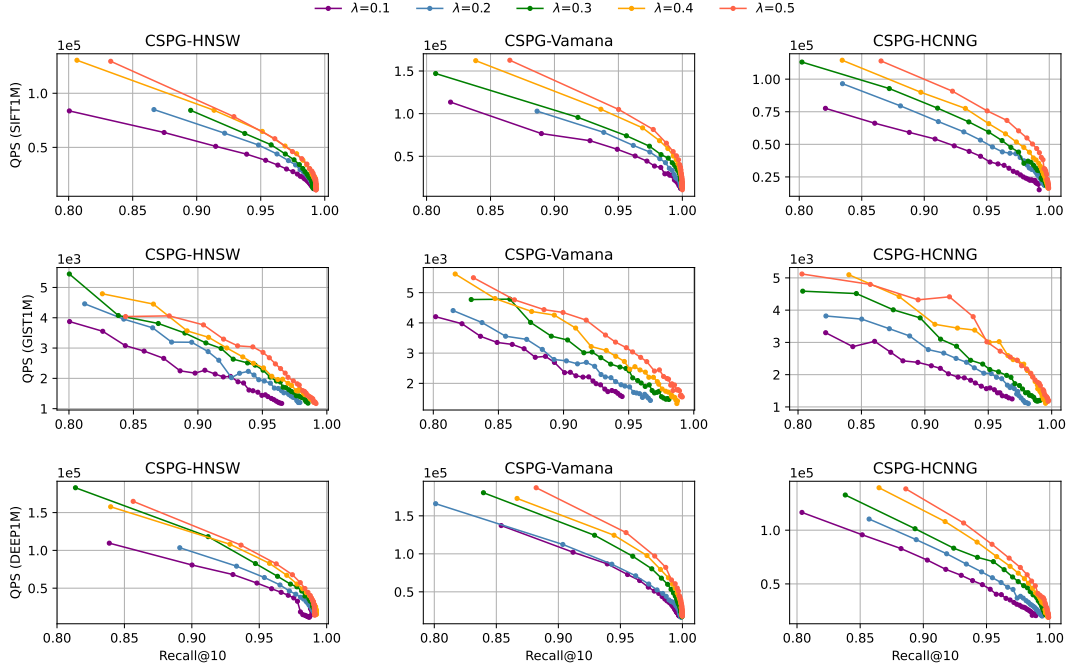

Figure 12: Query performance when varying the sampling ratio $\lambda$

Table 4: Hyperparameters selection sets

| Baseline | Parameters Selection Sets |
|---|---|
| *Vamana* | $R \in \{16, 32, 64\}, \alpha \in \{1.0, 1.2, 1.4\}, efConstruction \in \{64, 128, 256\}$ |
| *HNSW* | $M \in \{16, 32, 64\}, efConstruction \in \{64, 128, 256\}$ |
| *NSG* | $R \in \{16, 32, 64\}, efConstruction \in \{64, 128, 256\}$ |
| *HCNNG* | $s \in \{3, 5, 7\}, T \in \{5, 10, 15\}, Ls \in \{750, 1000, 1250\}$ |

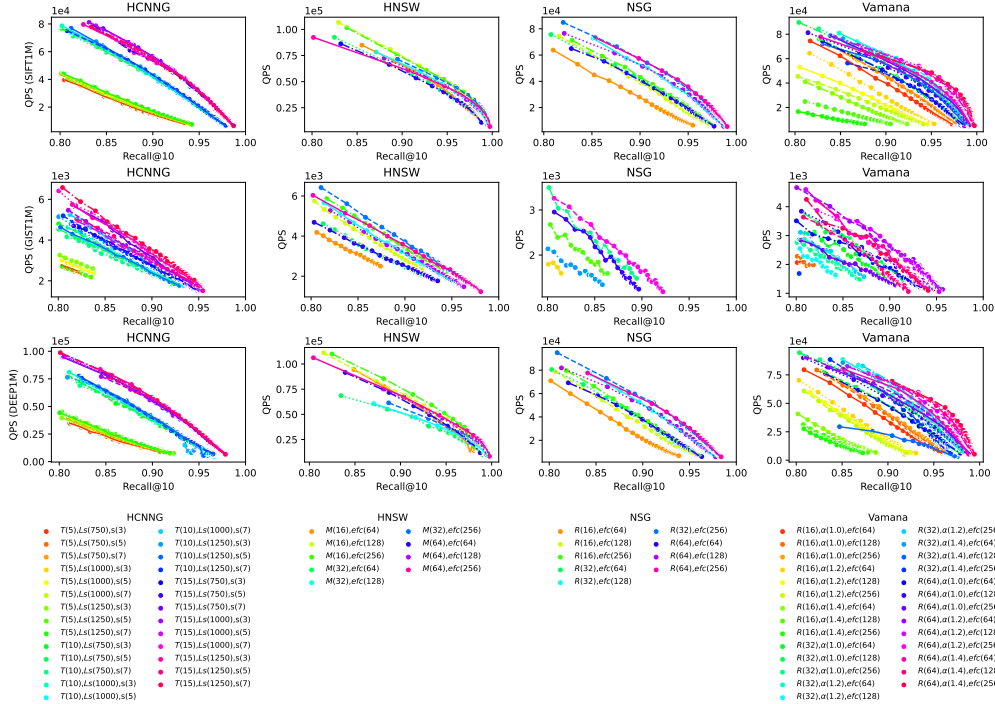

Figure 14: QPS v.s. Recall@10 over different parameters combination

